# Learning direction in global motion: two classes of psychophysically-motivated models

**V. Sundareswaran**      **Lucia M. Vaina***
Intelligent Systems Laboratory, College of Engineering,
Boston University
44 Cummington Street, Boston, MA 02215

## Abstract

Perceptual learning is defined as fast improvement in performance and retention of the learned ability over a period of time. In a set of psychophysical experiments we demonstrated that perceptual learning occurs for the discrimination of direction in stochastic motion stimuli. Here we model this learning using two approaches: a clustering model that learns to *accommodate* the motion noise, and an averaging model that learns to *ignore* the noise. Simulations of the models show performance similar to the psychophysical results.

## 1   Introduction

Global motion perception is critical to many visual tasks: to perceive self-motion, to identify objects in motion, to determine the structure of the environment, and to make judgements for safe navigation. In the presence of noise, as in random dot kinematograms, efficient extraction of global motion involves considerable spatial integration. Newsome and Colleagues (1989) showed that neurons in the macaque middle temporal area (MT) are motion direction-selective, and perform global integration of motion in their large receptive fields. Psychophysical studies in humans have characterized the limits of spatial and temporal integration in motion (Watamaniuk et. al, 1984) and the nature of the underlying motion computations (Vaina et. al 1990).

Since the psychophysical and neural substrate of global motion are fairly well understood, we were interested to see whether the perception of direction in such global motion stimuli can improve with practice. Studies specifically addressing this question for other early perceptual tasks have shown that improvements of performance obtained in the first experimental session are preserved in a subsequent session and retained over weeks. This is considered as perceptual learning (Gibson, 1953). Psychophysical studies of perceptual learning show that the beneficial effects of practice are lost if some stimulus parameters are changed significantly, such as orientation, spatial frequency or location in the visual field. Based on the time scale necessary for the improvement to occur, two major learning paradigms have been used in perceptual learning : slow, progressive learning (several thousand trials are required to reach stable performance) and fast learning (improvement occurs and stabilizes in the first 100-200 trials).

The idea of fast learning and the nature of its limits is attractive from a computational point of view because it encourages the exploration of practice-dependent plasticity found in the adult early visual system (Frégnac et. al., 1988, Gilbert and Wiesel 1992). A recent line of research in biologically motivated learning models originated by Poggio (Poggio, 1990) takes perceptual learning as evidence that "the brain may be able to synthesize–possibly in the cortex–appropriate task-specific modules that receive input from retinotopic cells and learn to solve the task after a short training phase in which they are exposed to examples of the task". Poggio and colleagues (1992) have illustrated this approach in learning vernier hyperacuity. Here, we adopted this general framework to study learning of direction in global motion. In contrast to Poggio et. al's supervised learning paradigm, we used unsupervised learning both in the psychophysical experiments and modeling of learning. We designed a set of psychophysical tasks to study whether fast learning occurs in discrimination of opposite directions of global motion and to explore the limits of this learning. To model the learning, we studied two models that differ in the way they deal with noise.

## 2   Psychophysics

Ball and Sekuler (1982, 1987) showed that discriminability of the direction of motion of two random dot patterns improved with training. In this learning paradigm, more than 2000 trials are required for reaching a stable performance. Such a "slow" learning time scale has been reported for the learning of other perceptual tasks, such as vernier acuity (McKee and Westheimer 1978, Fahle 1994), stereoacuity (Fendick and Westheimer, 1983; Ramachandran and Braddick 1973) and discrimination of line orientation (Vogels and Orban 1985). In contrast to this "slow learning," Fiorentini and Berardi (1981) showed that for learning the discrimination of complex gratings with two harmonics of different spatial phase, 100-200 trials suffice. Similarly Poggio et. al. (1992) show that a small number of trials suffice for significantly improving performance on a vernier hyperacuity task. Both studies discussed the specificity of learning to the stimulus attributes.

In our study, we used a two-alternative, forced-choice psychophysical procedure to measure the subject's ability to discriminate between two opposite directions of motion in dynamic random dot patterns in which 25% of the dots provide a correlated

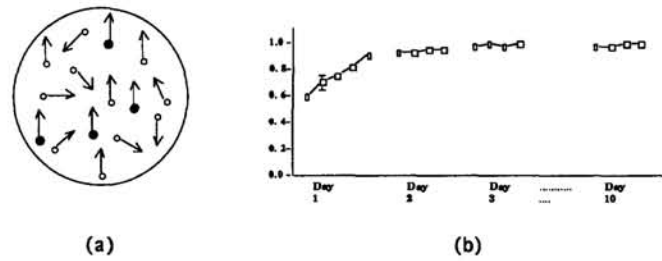

(a)                                      (b)

Figure 1: (a) Stimulus: a fraction of the dots (filled circles) move coherently in the signal direction; the rest (open circles) move randomly, (b) Fast improvement is observed on the first day of testing, and retained over a period of time. Each block consisted of 40 trials; data points averaged over 4 subjects, and errorbars show standard error.

motion signal spatially dispersed in a masking motion noise due to random motion of the reminder 75% of dots (Fig. 1(a)). Each trial lasted 90 msec during which two frames were presented (with inter-frame interval equal to zero). A session consisted of 4-6 blocks of 40 trials each. Feedback was not provided during the experimental sessions. Observers were required to maintain fixation on a fixation mark placed at $2^o$ from the imaginary circumference of the stimulus.

To investigate the effects of practice and their retention, the discrimination of leftward vs rightward direction of motion in the display was tested first on each experimental session for 3 consecutive days and repeated 10 days later. The results are presented in Fig. 1(b). For most observers, a fast and dramatic improvement was seen in the first day. Fig. 1(b) shows that learning was maintained in subsequent days, and even ten days later without any training in between. Examination of the individual observers' data revealed that improvement of performance occurred only if they started above chance level. This suggests that this fast learning might imply the improvement of an existing representation of the stimulus.

In additional experiments, we did not find transfer to another direction of motion (up/down), indicating that the learning is selective to specific characteristics of the stimulus. Details of experiments testing the limits of the learning appear in Vaina et. al (1995).

## 3   Modeling

We propose two paradigms to model the learning found in psychophysics. Both use directionally-tuned units with properties similar to those of neurons found in MT (Maunsell and Van Essen, 1983). Schematic MT neurons used in our modeling integrate global information by summing over localized responses:

$$x_t = \sum_{i=1}^{n} e^{-(\frac{1}{2\sigma_h^2}(\theta_i - \theta_t)^2)}. \tag{1}$$

where $\sigma_h$ is the standard deviation of the tuning, and information from $n$ local responses is taken into account. The responses of a collection of such units (each

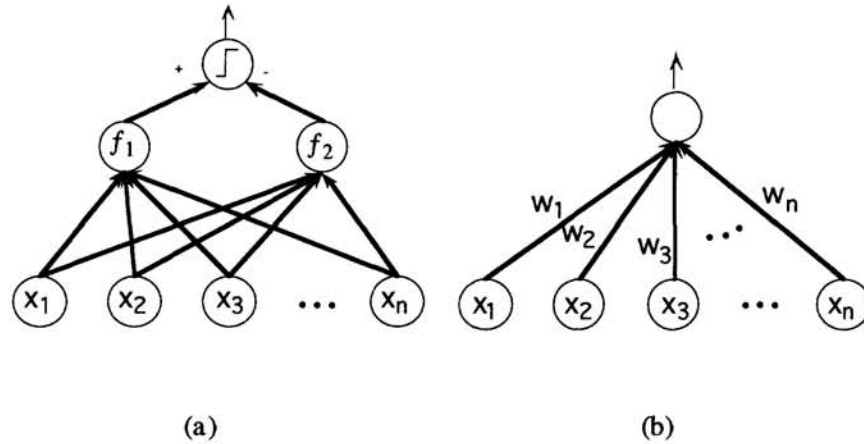

(a)                                                    (b)

Figure 2: Architectures of the models: (a) Learning to Accommodate: Cluster Gaussians $f_1$ and $f_2$ operate on input vector **X** to decide global motion direction, (b) Learning to Ignore: Global motion direction is computed as a weighted combination of units' preferred directions.

tuned to a different direction) form the input vector to the models.

## 3.1   Learning to accommodate

This model is based on grouping *similar* data. In this paradigm, after learning, the network has an estimate of the contribution of noise, and takes it into account when performing the task of discriminating between two different directions of motion; so, we say that the model "learns to accommodate" the noise.

Fig 2(a) contains a schematic of this model. The representation vector consists of responses $x_1.x_2, \ldots, x_n$ of the directionally-tuned units.. Clustering is done in the space of the representation vectors. The model is a combination of HyperBF-like functions and clustering (Poggio et. al, 1992, Moody and Darken 1989). We use gaussians with mean at the cluster centers, and "move" the cluster centers by a learning algorithm.

A cluster gaussian computes a gaussian function of the representation vector for the current stimulus from the current center of the corresponding cluster. We say "current" because the center is moved as the learning proceeds. At any given point, a center is at the current best estimate of the center of the corresponding data cluster. More precisely, the learning rule to modify the jth coordinate of the center is given by:

$$c_{w,j}^{(t+1)} = c_{w,j}^{(t)} + \eta * (x_j^{(t)} - c_{w,j}^{(t)}), \tag{2}$$

where $w$ is the index of "winning" cluster and $c_{w,j}$ is the $j$th coordinate of the center for the $w$th cluster. This moves the center towards the new data vector $X$ that has been judged to belong to the $w$th cluster. The parameter $\eta$ controls the learning rate.

## 3.2 Learning to ignore

The learning involved in this approach is Hebbian, and is termed "learning to ignore," because in this weighted averaging scheme, as learning occurs, the weights for the noise response are progressively reduced to zero, leaving only the contribution from the signal. In other words, the network learns to ignore the noise (Vaina et. al 1995).

The model's output is a global motion direction. If the weights associated with the responses $x_1, x_2, \ldots x_n$ are $w_1, w_2, \ldots w_n$, the global motion direction is calculated as

$$\theta_o = \tan^{-1}\left(\frac{\sum w_i X_i \sin\theta_i}{\sum w_i X_i \cos\theta_i}\right),$$

where $t_i$ is the tuned direction (angle) of the $i$th unit. A schematic of the model is shown in Fig. 2(b). The global direction is judged to be rightward if $+\theta_t > \theta_o > -\theta_t$.

We have examined two different learning rules: exposure-based learning, and self-supervised learning.

**Exposure-based learning:** The weight corresponding to a unit is incremented by an amount proportional to the current weight. Only units whose response values are above a certain threshold are allowed to increase their weights. This learning rule favors units that fire consistently:

$$w_i \leftarrow w_i + \eta w_i, \text{ if } x_i > r_t, \tag{3}$$

where $r_t$ is a threshold, and $\eta$ is a small fraction that controls the learning rate.

**Self-supervised learning:** The weight corresponding to a unit is increased by an amount proportional to the product of the current weight and a decreasing function (exponential) of the angular difference between the calculated global motion direction and the direction of tuning of the unit:

$$w_i \leftarrow w_i + \eta w_i e^{(-(\theta_i - \theta_o)^2/2\sigma_t)}, \text{ if } x_i > r_t.$$

In this case, the model uses its own estimate of the global direction as an internal feedback to determine the learning.

An approach similar to this model for learning vernier hyperacuity was proposed by Weiss et. al (1993).

## 4 Experiments

For the simulations, the input (motion) vectors were represented by their angles relative to the positive horizontal axis (i.e., the magnitude is ignored). The responses of the directionally-tuned units can be directly computed from the angles (see Eqn. 1; alternatively, cosine tuning functions were used, and similar results were obtained). For each trial, the coherent motion direction was randomly decided. In the experiments, $\sigma_h$ was set to $\frac{\pi}{n_d}$, where $n_d$ is the number of unit preferred directions; the directions $\theta_t$ are chosen by uniformly dividing $n_d$ in to $2\pi$. In all the experiments, eight preferred directions were used; each trial contained 40 random dots moving with a correlation of 25% (the same correlation as in the tests with human subjects);

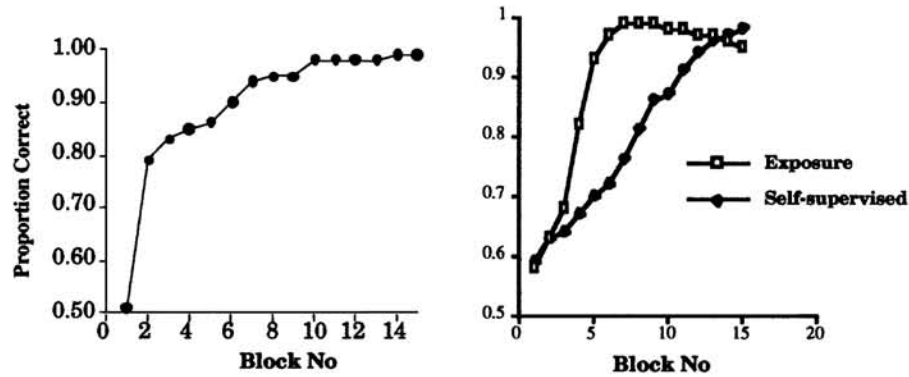

Figure 3: Results from typical simulation runs. On the left is the curve for the *learning to accommodate* model ($\eta = 0.005$; cluster $\sigma = 0.5$) and on the right are the curves for the *learning to ignore* model.

each block consisted of 50 trials. The performance is measured by *fraction correct*, corresponding to the fraction of the inputs that were correctly classified.

For both models, we did simulations to justify the architecture of the models by disabling learning, and studying the performance of the models for increasing correlation. The performance was qualitatively similar to that described in perceptual studies of these stimuli in humans and monkeys.

Learning curves from simulations of both models are shown in Fig. 3. These are results from averaging over the performance in ten simulation runs. As can be seen, the models learn to do the discrimination. While both models successfully learned to do the direction discrimination, there are quantitative differences in the learning curves. The *learning to accommodate* paradigm improved very rapidly (somewhat faster than the human subjects). The *exposure-based* learning rule for the *learning to ignore* paradigm learned at a rate comparable with the human observers. However, the final performance in this case was not very stable, and oscillated (not shown), consistent with the observations of Weiss et. al (1993) in learning vernier hyperacuity. The *self-supervised* rule, on the other hand, reached a stable level of performance, but the learning was slower; the reason is that this learning rule exhibits instability if a high value of learning rate ($\eta$) is used.

If either model trained on inputs containing horizontal correlated motion was later presented with inputs containing vertical correlated motion, the performance dropped to pre-training levels. This non-transfer is consistent with psychophysical results mentioned in Section 2.

## 5   Discussion

In this study, we focused on learning of global motion direction, and not on motion perception. Motion perception is well-understood both physiologically and psy-

chophysically. In a series of studies of neuronal correlates of the perceptual decision of direction of motion in stochastic motion signals like the stimuli used here, together with studies of performance on these stimuli on monkeys with MT lesions, Newsome and his collaborators (for a review, Newsome et. al, 1989) provide strong support for the hypothesis that perceptual judgements of motion direction are by and large based on the directional signals carried by MT neurons.

Salzman and Newsome (1994) reported that in trained monkeys, the perception of motion in stochastic noise is more likely mediated by a winner-take-all mechanism than by a weighted averaging mechanism. Interestingly, our clustering model, after learning, is behaviorally similar to the winner-take-all mechanism: the dominating component of the representation results in the assignment to the closest cluster.

Our psychophysical studies clearly demonstrate that perceptual learning of global motion direction occurs. While significant progress has been made to understand the mechanisms and underlying circuitry of learning (Zohary et. al 1994) as yet there is no satisfactory biologically explanation of how and where this learning may occur (Zohary and Newsome 1994).

Our focus in this paper was on computational models of learning direction in global motion. For this, we proposed two approaches, which differ in the way they deal with noise: one learns to accommodates noise, and the other learns to ignore it. We do not advocate that one or the other of the models we proposed here provides the biologically correct choice for this task. However, together with the psychophysics described here these models suggest new experiments which we are now exploring both psychophysically and computationally.

We hope that by closely connecting models and psychophysics while keeping in mind the aim of neuronal compatibility, we will make progress in understanding how the cortex learns so fast to discriminate direction of motion in extremely noisy situations.

## Acknowledgments

This research was conducted at the Intelligent Systems Laboratory of Boston University, College of Engineering. LMV and VS were supported in part by grants from the Office of Naval Research (# N00014-93-1-0381) and the National Institute of Health (EY RO1- 07861) to LMV. VS was in part supported by the Boston University College of Engineering Dean's postdoctoral fellowship. We gratefully acknowledge additional financial support for VS from the Dean's special research fund.

## Footnotes

*Please address all correspondence to Lucia Vaina

## References

[1] K. Ball and R. Sekuler. Direction-specific improvement in motion discrimination. *Vision Research*, 27(6):953–965, 1987.

[2] M. Fahle. Human pattern recognition: parallel processing and perceptual learning. *Perception*, 23:411–427, 1994.

[3] M. Fendick and G. Westheimer. Effects of practice and the separation of test targets on foveal and perifoveal hyperacuity. *Vision Research*, 23:145–150, 1983.

[4] A. Fiorentini and N. Berardi. Perceptual learning specific for orientation and spatial frequency. *Nature*, 287(5777):43–44, September 1980.

[5] Y. Frégnac, D. Shulz, S. Thorpe, and E. Bienstock. A cellular analogue of visual cortical plasticity. *Nature*, 333:367–370, 1988.

[6] E. J. Gibson. Improvements in perceptual judgement as a function of controlled practice of training. *Psychology bulletin*, 50:402–431, 1953.

[7] C. D. Gilbert and T. N. Wiesel. Receptive field dynamics in adult primary visual cortex. *Nature*, 356:150–152, 1992.

[8] A. Karni and D. Sagi. Where practice makes perfect in texture discrimination: Evidence for primary visual cortex plasticity. *Proc. Natl. Acad. Sci. USA*, 88:4966–4970, June 1991.

[9] J. H. R. Maunsell and D. C. Van Essen. Functional properties of neurons in the middle temporal visual area of the macaque monkey i: selectivity for stimulus direction, speed, and orientation. *J. Neurophysiology*, 49:1127–1147, 1983.

[10] S. P. McKee and G. Westheimer. Improvement in venier acuity with practice. *Perception & Psychophysics*, 24:258–262, 1978.

[11] W. T. Newsome, K. H. Britten, and J. A. Movshon. Neuronal correlates of a perceptual decision. *Nature*, 341:52–54, 1989.

[12] T. Poggio. A theory of how the brain might work. In *Cold Spring Harbor Symposia on Quantitative Biology*, pages 899–910. Cold Spring Harbor Laboratory Press, 1990.

[13] T. Poggio, M. Fahle, and S. Edelman. Fast perceptual learning in visual hyperacuity. *Science*, 256:1018–1021, 1992.

[14] T. Poggio and F. Girosi. A theory of networks for approximation and learning. AI Memo 1140, M.I.T, July 1989.

[15] V. S. Ramachandran and O. Braddick. Orientation-specific learning in stereopsis. *Perception*, 2:371–376, 1973.

[16] C. D. Salzman and W. T. Newsome. Neural mechanisms for forming a perceptual decision. *Science*, 264:231–237, 1994.

[17] L. M. Vaina, V. Sundareswaran, and J. Harris. Computational learning and natural learning. Cognitive Brain Research, 1995. (in press).

[18] L. M. Vaina, N. M. Grzywacz, and M. LeMay. Structure from motion with impaired local-speed and global motion-field computations. *Neural Computation*, 2:420–435, 1990.

[19] R. Vogels and G. A. Orban. The effect of practice on the oblique effect in line orientation judgements. *Vision Research*, 25:1679–1687, 1985.

[20] S. Watamaniuk, R. Sekuler, and D. Williams. Direction perception in complex dynamic displays: the integration of direction information. *Vision Research*, 24:55–62, 1984.

[21] Y. Weiss, S. Edelman, and M. Fahle. Models of perceptual learning in vernier hyperacuity. *Neural Computation*, 5:695–718, 1993.

[22] E. Zohary, S. Celebrini, K. H. Britten, and W. T. Newsome. Neuronal plasticity that underlies improvement in perceptual performance. *Science*, 263:1289–1292, March 1994.

[23] E. Zohary and W. T. Newsome. Perceptual learning in a direction discrimination task is not based upon enhanced neuronal sensitivity in the sts. *Investigative Ophthalmology and Visual Science supplement*, page 1663, 1994.
